# Accounting for network effects in neuronal responses using L1 regularized point process models

**Ryan C. Kelly**[*]
Computer Science Department
Center for the Neural Basis of Cognition
Carnegie Mellon University
Pittsburgh, PA 15213
rkelly@cs.cmu.edu

**Matthew A. Smith**
University of Pittsburgh
Center for the Neural Basis of Cognition
Pittsburgh, PA 15213
masmith@cnbc.cmu.edu

**Robert E. Kass**
Department of Statistics
Center for the Neural Basis of Cognition
Machine Learning Department
Carnegie Mellon University
Pittsburgh, PA 15213
kass@stat.cmu.edu

**Tai Sing Lee**
Computer Science Department
Center for the Neural Basis of Cognition
Carnegie Mellon University
Pittsburgh, PA 15213
tai@cnbc.cmu.edu

## Abstract

Activity of a neuron, even in the early sensory areas, is not simply a function of its local receptive field or tuning properties, but depends on global context of the stimulus, as well as the neural context. This suggests the activity of the surrounding neurons and global brain states can exert considerable influence on the activity of a neuron. In this paper we implemented an L1 regularized point process model to assess the contribution of multiple factors to the firing rate of many individual units recorded simultaneously from V1 with a 96-electrode "Utah" array. We found that the spikes of surrounding neurons indeed provide strong predictions of a neuron's response, in addition to the neuron's receptive field transfer function. We also found that the same spikes could be accounted for with the local field potentials, a surrogate measure of global network states. This work shows that accounting for network fluctuations can improve estimates of single trial firing rate and stimulus-response transfer functions.

## 1 Introduction

One of the most striking features of spike trains is their variability – that is, the same visual stimulus does not elicit the same spike pattern on repeated presentations. This variability is often considered to be "noise," meaning that it is due to unknown factors. Identifying these unknowns should enable better characterization of neural responses. In the retina, it has recently become possible to record from a nearly complete population of certain types of ganglion cells in a region and identify the

_________________________

[*]Data was collected by RCK, MAS and Adam Kohn in his laboratory as a part of a collaborative effort between the Kohn laboratory at Albert Einstein College of Medicine and the Lee laboratory at Carnegie Mellon University. This work was supported by a National Science Foundation (NSF) Integrative Graduate Education and Research Traineeship to RCK (DGE-0549352), National Eye Institute (NEI) grant EY018894 to MAS, NSF 0635257 and NSF CISE IIS 0713206 to TSL, NIMH grant MH064537 to REK, and NEI grant EY016774 to Adam Kohn. We thank Adam Kohn for collaboration, and we are also grateful to Amin Zandvakili, Xiaoxuan Jia and Stephanie Wissig for assistance in data collection. We also thank Ben Poole for helpful comments.

correlation structure of this population [1]. However, in cerebral cortex, recording a full population of individual neurons in a region is currently impossible, and large scale recordings *in vivo* have been rare. Cross-trial variability is often removed in order to better reveal the effect of a signal of interest. Classical methods attempt to explain the activity of neurons only in terms of stimulus filters or kernels, ignoring sources unrelated to the stimulus.

An increasing number of groups have modeled spiking with point process models [2, 3, 4] to assess the relative contributions of specific sources. Pillow et al.[3] used these methods to model retinal ganglion cells, and they showed that the responses of cells could be predicted to a large extent using the activity of nearby cells. We apply this technique to model spike trains in macaque V1 *in vivo* using L1 regularized point process models, which for discrete time become Generalized Linear Models (GLMs) [5]. In addition to incorporating the spike trains of nearby cells, we incorporated a meaningful summary of local network activity, the local field potential (LFP), and show that it also can explain an important part of the neuronal variability.

## 2    L1 regularized Poisson regression

Fitting an unregularized point process model or GLM is simple with any convex optimization method, but the kind of neural data we have collected typically has a likelihood function that is relatively flat near its minimum. This is a data constraint: there simply are not enough spikes to locate the true parameters. To solve this over-fitting problem, we take the approach of regularizing the GLMs with an L1 penalty (Lasso) on the log-likelihood function. Here we provide some details of how we fit L1-regularized GLMs using a Poisson noise assumption on data with large dimensionality. In general, a point process may be represented in terms of a conditional intensity function and, assuming the data (the spike times) are in sufficiently small time bins, the resulting likelihood function may be approximated by a Poisson regression likelihood function. For ease of notation we leave the spiking history and other covariates implicit and write the conditional intensity (firing rate) at time $t$ as $\mu(t)$. We then model the log of $\mu(t)$ as a linear summation of other factors:

$$\log \mu(t) = \sum_{j}^{N} \theta_j v_j^{(t)} = \theta V^{(t)} \tag{1}$$

where $v_j$ is a feature of the data and $\theta_j$ is the corresponding parameter to be fit, and $\theta = \{\theta_1, .., \theta_N\}$. We define $V$ to be a $N \times T$ matrix ($N$ parameters, $T$ time steps) of variables we believe can impact the firing rate of a cell, where each column $V^{(t)}$ of $V$ is $v_1^{(t)}, ..., v_N^{(t)}$, which are the collection of observables, including input stimulus and measured neural responses.

We define $y = y_1...y_T$, with $y_t \in \{0, 1\}$ as the observed binary spike train for the cell being modeled, and let $\mu_t = \mu(t)$. The likelihood of the entire spike train is given by:

$$P(Y = y_1...y_T) = \prod_{t}^{T} \frac{(\mu_t)^{y_t} \exp(-\mu_t)}{y_t!} \tag{2}$$

We obtain the log-likelihood by substituting Equation 1 into Equation 2 and taking the log:

$$L(\theta) = \sum_{t}^{T} (y_t \theta V^{(t)} - \exp(\theta V^{(t)}) - \log y_t!) \tag{3}$$

Maximizing the likelihood with L1 penalty is equivalent to finding the $\theta$ that minimizes the following cost function:

$$R = -L(\theta) + \sum_{j=1}^{N} \lambda_j |\theta_j| \tag{4}$$

An L1 penalty term drives many of the $\theta_i$ coefficients to zero. Fitting this equation with an L1 constraint is computationally difficult, because many standard convex optimization algorithms are only guaranteed to converge for differentiable functions. Friedman et al. [5] discuss how coordinate descent can efficiently facilitate GLM fitting on functions with L1 penalties, and they provide a derivation for the logistic regression case. Here we show a derivation for the Poisson regression case.

We approximate $L(\theta)$ with $L_Q(\theta)$, a quadratic Taylor series expansion around the current estimate $\tilde{\theta}$. Then we proceed to minimize $R_Q = -L_Q(\theta) + \sum_{j=1}^{N} \lambda_j |\theta_j|$.

Given $\tilde{\theta}$, we can compute $\tilde{\mu}$, the current estimate of $\mu$. A coordinate descent step for coordinate $j$ amounts to the minimization of $R_Q$ with respect to $\theta_j$, for $j \in 1 \ldots N$.

$$\text{For } \tilde{\theta}_j > 0, \frac{dR_Q}{d\theta_j} = \omega_j + \theta_j \sum_t^T \tilde{\mu}_t (v_j^{(t)})^2 + \lambda_j, \qquad \text{For } \tilde{\theta}_j < 0, \frac{dR_Q}{d\theta_j} = \omega_j + \theta_j \sum_t^T \tilde{\mu}_t (v_j^{(t)})^2 - \lambda_j$$

$$\text{where } \omega_j = \sum_t^T v_j^{(t)} \left( -y_t + \tilde{\mu}_t - \tilde{\mu}_t (v_j^{(t)} \tilde{\theta}_j) \right) \tag{5}$$

This is a linear function with positive slope, and a discontinuity at $\theta_j = 0$. If $-\lambda_j < \omega_j < \lambda_j$, $\frac{dR_Q}{d\theta_j} \neq 0$ and the minimum is at this discontinuity, $\theta_j = 0$. Otherwise, if $|\omega_j| \geq \lambda_j$, $\frac{dR_Q}{d\theta_j} = 0$ when

$$\theta_j = -(\omega_j - \lambda_j)/(\sum_t^T \tilde{\mu}_t (v_j^{(t)})^2), \quad \text{for } \omega_j \geq \lambda_j \tag{6}$$

$$\theta_j = -(\omega_j + \lambda_j)/(\sum_t^T \tilde{\mu}_t (v_j^{(t)})^2), \quad \text{for } \omega_j \leq -\lambda_j \tag{7}$$

We cyclically repeat these steps on all parameters until convergence.

## 2.1 Regularization path

To choose efficiently a penalty that avoids over-fitting, we implement a regularization path algorithm [6, 5]. The algorithm proceeds by computing a sequence of solutions $\theta^{(1)}, \theta^{(2)} \ldots \theta^{(L)}$ for $\lambda^{(1)}, \lambda^{(2)} \ldots \lambda^{(L)}$. We standardize $V$ (i.e. make each observable have mean 0 and standard deviation 1) and include a constant term $v_1$, which is not penalized. With this normalization, we set all $\lambda_j$ equal to the same $\lambda$, except there is no penalty for $v_1$.

In the coordinate descent method, we start with a $\lambda^{(1)} = \lambda_{\max} = \max_j |\omega_j|$, which is large enough so that all coefficients are dominated by the regularization, and hence all coefficients are 0 for this heavy penalty. In determining $\lambda_{\max}$, $\omega_j$ is computed based on the constant term $v_1$ only. Initially, the active set $A^{(1)}$ is empty, because $\lambda > \lambda_{\max}$. The active set is the set of all coordinates with non-zero coefficients for which the coordinate descent is being performed. As $\lambda$ is reduced and becomes smaller than $\lambda_{\max}$, more and more non-zero terms will be included in the active set. For step $i$, we compute the solution $\theta^{(i)}$ using penalty $\lambda^{(i)}$ and $\theta^{(i-1)}$ as a warm start. As the regularization parameter $\lambda$ is decreased, the fitted models begin by under-fitting the data (with large $\lambda$) and progress through the regularization path to over-fitting (with small $\lambda$). The above algorithm works much faster when the active set is smaller, and we can halt the algorithm before over-fitting occurs.

The purpose of this regularization path is to find the best $\lambda$. To quantitatively assess the model fits, we employ an ROC procedure [7]. To compute the ROC curve based on the conditional intensity function $\mu(t)$, we first create a thresholded version of $\mu(t)$ which serves as the prediction of spiking:

$$\hat{r}_c(t) = 1 \text{ if } \mu(t) \geq c \tag{8}$$

$$0 \text{ if } \mu(t) < c \tag{9}$$

For each fixed threshold $c$, a point on the ROC curve is the true positive rate (TPR) versus the false positive rate (FPR). At each $\lambda$ in the regularization path, we compute the area under the ROC curve (AUC) to assess the relative performance of models fit below using a 10-fold cross validation procedure. An alternative and natural metric is the likelihood value, and the peak of the regularization path was very similar between AUC and likelihood. We focus on AUC results because it was easier to relate the AUCs from different cells, some of which had very different likelihood values.

## 3 Modeling neural data

We report results from the application of Eq. (4) to neural data. The models here contain combinations of stimulus effects (spatio-temporal receptive fields), coupling effects (history terms and past

spikes from other cells), and network effects (given by the LFP). We find that cells had different degrees of contributions from the different terms, ranging from entirely stimulus-dependent cells to entirely network-dependent cells.

## 3.1 Methods

The details of the array insertion have been described elsewhere [8]. Briefly, we inserted the array 0.6 mm into cortex using a pneumatic insertion device [9], which led to recordings confined mostly to layers 2–3 of parafoveal V1 (receptive fields within 5° of the fovea) in an anesthetized and paralyzed macaque (sufentanil anesthesia). Signals from each microelectrode were amplified and bandpass filtered (250 Hz to 7.5 kHz) to acquire spiking data. Waveform segments that exceeded a threshold (set as a multiple of the rms noise on each channel) were digitized (30 kHz) and sorted off-line. We first performed a principal components analysis by waveform shape [10] and then refined the output by hand with custom time-amplitude window discrimination software (written in MATLAB; MathWorks). We studied the responses of cells to visual stimuli, presented on a computer screen. All stimuli were generated with custom software on a Silicon Graphics Octane2 Workstation and displayed at a resolution of $1024 \times 768$ pixels and frame rate of 100 Hz on a CRT monitor (stimulus intensities were linearized in luminance). We presented Gaussian white noise movies, with 8 pixel spatial blocks chosen independently from a Gaussian distribution. The movies were 5° in width and height, 320 by 320 pixels. The stimuli were all surrounded by a gray field of average luminance. Frames lasted 4 monitor refreshes, so the duration of each frame of noise was 40 ms. The average noise correlation between pairs of cells was 0.256.

The biggest obstacle for fitting models is the huge dimensionality in the number of parameters and in the large number of observations. To reduce the problem size, we binned the spiking observations at 10 ms instead of 1 ms. The procedures we used to reduce the parameter sizes are given in the corresponding sections below. We used cross validation to estimate the performance of the models on 10 different test sets. Each test set consisted of 12,000 test observations and 180,000 training observations. The penalty in the regularization path with the largest average area across all the cross validation runs was considered the optimal penalty.

The full model $\mu(t) = \mu_{\text{STIM}} + \mu_{\text{COUP}} + \mu_{\text{LFP}}$ has the following form:

$$\log \mu(t) = \sum_x \sum_y \sum_\tau k_{xy\tau} s_{xy}(t-\tau) + \sum_i^M \sum_{\tau=1}^{100} \gamma_i r_i(t-\tau) + \sum_i^E \beta_i x_i(t) \qquad (10)$$

## 3.2 Stimulus effects

For modeling the stimulus alone we used the form

$$\log \mu_{\text{STIM}}(t) = \sum_x \sum_y \sum_\tau k_{xy\tau} s_{xy}(t-\tau) \qquad (11)$$

Here, $s_{xy}(t-\tau)$ is an individual feature of the stimulus $\tau$ ms before the current observation (time $t$). If we were to use pixel intensities over the last 150 ms (15 observations), the $320 \times 320$ movie would have $1\,536\,000$ parameters, a number far too large for the fitting method and data. We took the approach of first restricting the movie to a much smaller region (40x40 pixels) chosen using spike-triggered average (STA) maps of the neural responses. Then, we transformed the stimulus space with overlapping Gaussian bump filters, which are very similar to basis functions. The separation of the bump was 4 pixels spatially in the 40x40 pixel space, and 2 time points (20 ms). The total number of parameters was $10 \times 10 \times 7 = 700$, which is 100 parameters for each of 7 distinct time points. Thus, $s_{xy}(t-\tau)$ corresponds to the convolution of a small Gaussian bump indexed by $x, y, \tau$ with the recent stimulus frames. Figure 1 shows the regularization path for one example cell.

For each model (11), we chose the $\lambda$ corresponding to the peak of the regularization path. Figure 2A shows the $k$ parameters for some example cells transformed back to the original pixel space, with the corresponding STAs alongside for comparison. The models produce cleaner receptive fields, a consequence of the L1 regularization. Figure 2D shows the population results for these models. The distribution of AUC values is generally low, with many cells near chance (.5), and a smaller portion of cells climbing to 0.6 or higher. This suggests that a linear receptive field may not be appropriate

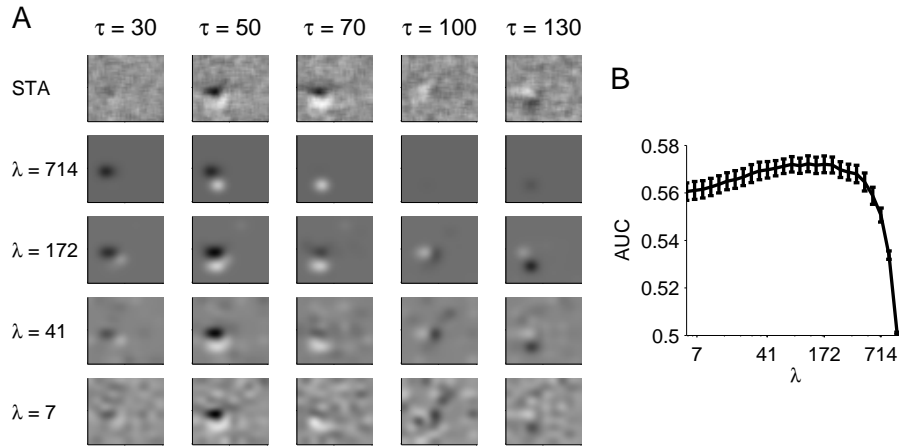

Figure 1: Example of fitting a GLM with stimulus terms for a single cell. A: For four L1 penalties ($\lambda$), the corresponding $\{k_i\}$ are shown, with the STA above for reference. For high $\lambda$, the model is sparser. B: The regularization path for this same cell. $\lambda = 172$ is the peak of the AUC curve and is thus the best model by this metric.

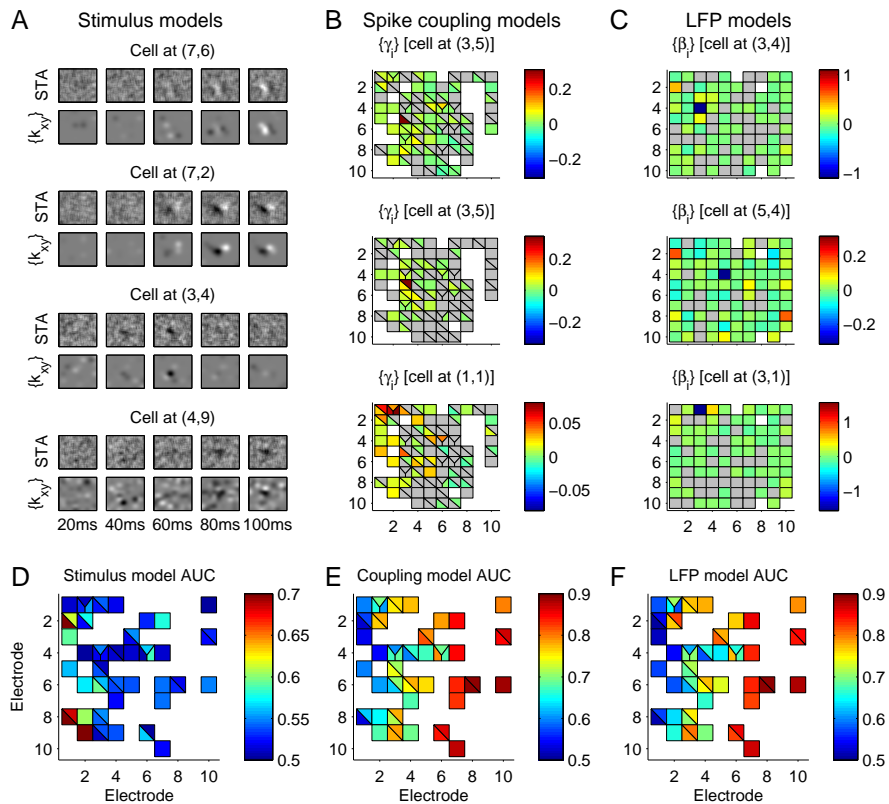

Figure 2: Different GLM types. A: 4 example stimulus models, with the STAs shown for reference. These models correspond to the AUC peaks of their respective regularization paths. B: 3 example cells fit with spike coupling models. The coefficients are shown with respect to the cell location on the array. If multiple cells were isolated on the same electrode, the square is divided into 2 or 3 parts. Nearby electrodes tend to have more strength in their fitted coefficients. C: 3 example cells fit with LFP models. As in B, nearby electrodes carry more information about spiking. D-F: Population results for A-C. These are plots of the AUCs for the 57 cells modeled.

for many of these cells. In addition, there is an effect of electrode location, with cells with the highest AUC located on the left side of the array.

## 3.3 Spike coupling effects

For the coupling terms, we used the history of firing for the other cells recording in the array as well as the history for the cell being modeled. These take the form:

$$\log \mu_{\text{COUP}}(t) = \sum_{i}^{M} \sum_{\tau=1}^{100} \gamma_i r_i(t - \tau) \tag{12}$$

with $\gamma_i$ being the coupling strength/coefficient, and $r_i(t - \tau)$ being the activity of the $i^{\text{th}}$ neuron $\tau$ ms earlier, and $M$ being the number of neurons. Thus the influence from a surrounding neuron is computed based on its spike count in the last 100ms. As expected, nearby cells generally had the largest coefficients (Figure 2B), indicating that cells in closer proximity tend to have more correlation in their spike trains. We observed a large range of AUC values for these fits (Figure 2E), from near chance levels up to .9. There was a significant ($p < 10^{-6}$) negative correlation between the AUC and the number of nonzero coefficients used in the model. Thus, the units which were well predicted by the other firing in the population also did not require a large number of parameters to achieve the best AUC possible. Also apparent in the figure is that the relationship between spike train predictability and array location had the opposite pattern of the stimulus model results, with units toward the left side of the array generally having smaller AUCs based on the population activity than units on the right side.

The models described above had one parameter per cell in the population, with each parameter corresponding to the firing over a 100 ms past window. We also fit models with 3 parameters per cell in the population, corresponding to the spikes in three non-overlapping temporal epochs (1-20 ms, 21-50 ms, 51-100 ms). These were considered to be independent parameters, and thus the active set could contain none, some, or all of these 3 parameters for each cell. The mean AUC across the population was .01 larger with this increased parameter set, but also the mean active set size was 100 elements larger. We did not attempt to model effects on very short timescales, since we binned the spikes at 10 ms.

## 3.4 Network models

The spiking of cells in the population serves to help predict spiking very well for many cells, but the cause of this relationship remains undetermined. The specific timing of spikes may play a large role in predicting spikes, but alternatively the general network fluctuations could be the primary cause. To disentangle these possibilities, we can model the network state using the LFP as an estimate:

$$\log \mu_{\text{LFP}}(t) = \sum_{i}^{E} \beta_i x_i(t) \tag{13}$$

Here, E is the number of surrounding electrodes, $x_i$ is the LFP value from electrode $i$, and $\beta_i$ is the coefficient of the LFP influence on the spiking activity of the neuron being considered. Figure 2C shows the model coefficients of several cells when $\{x_i\}$ are the LFP values at time $t$. The variance in the coefficient values falls off with increasing distance, with distant electrodes providing relatively less information about spiking. Across the population, the AUC values for the cells are almost the same as in the spike coupling models (Figure 2F), and consequently the spatial pattern of AUC on the array is almost identical. We also investigated models built using the LFP power in different frequency bands, and we found that the LFP power in the gamma frequency range (30-80Hz) produced similar results. With these models, the AUC distributions were remarkably similar to the models built with spike coupling terms (Figure 2E). The LFP reflects activity over a very broad region, and thus for these data the connectivity between most pairs in the population do not generally have much more predictive power than the more broad network dynamics. This suggests that much of the power of the spike coupling terms above is a direct result of both cells being driven by the underlying network dynamics, rather than by a direct connection between the two cells unrelated

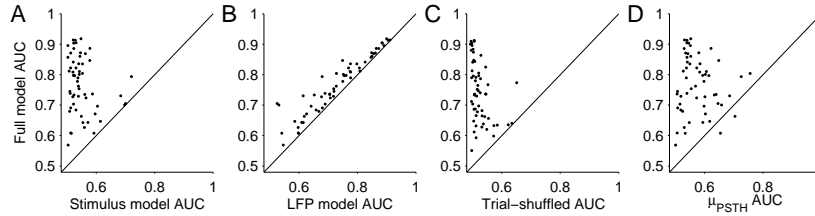

Figure 3: Scatter plots of the AUC values for the population under different models and conditions. A,B: The full model improves upon the individual LFP or stimulus models. C: For most cells, trial shuffling the spike trains destroys the effectiveness of the models. D: Taking the network state and cell spikes into account generally yields a larger AUC than $\mu_{\text{PSTH}}$.

to the more global dynamics. Models of spike coupling with more precise timing ($< 10$ ms) may reflect information that these LFP terms would fail to capture.

## 4   Capturing variability and predicting the PSTH

Neuronal firing has long been accepted to have sources of noise that have typically been ignored or removed. The simplest conception is that each of these cells has an independent source of intrinsic noise, and to recover the underlying firing rate function we can simply repeat a stimulus many times. We have shown above that for many cells, a portion of the noise is not independent from the rest of the network and is related to other cells and the LFP. The population included a distribution of cells, and the GLMs showed that some cells included mostly network terms, and other cells included mostly stimulus terms. For most cells, the models included significant contributions from both types of terms.

From Figure 3A and 3B we can see that the inclusion of network terms does indeed explain more of the spikes than the stimulus model alone. It is theoretically possible that the LFP or spikes from other cells are reflecting higher order terms of the stimulus-response relationship that the linear model fails to capture, and the GLM is harnessing these effects to increase AUC. We performed an AUC analysis on test data from the same neurons: 120 trials of the same 30 second noise movie. Since the stimulus was repeated we were able to shuffle trials. Any stimulus information is present on every trial of this repeated stimulus, and so if the AUC improvement is entirely due to the network terms capturing stimulus information, there should be no decrease in AUC in the trial-shuffled condition. Figure 3C shows that this is not the case: trial shuffling reduces AUC values greatly across the population. This means that the network terms are not merely capturing extraneous stimulus effects.

Kelly et al. [11] show that when taking the network state into account with a very simple GLM, the signal to noise in the stimulus-response relationship was improved. The PSTH is typically used as a proxy for the stimulus effects. The idea is that any noise terms are averaged out after many trials to the same repeated stimulus. For the data set of a single repeated noise movie, we made a comparison of the AUC values computed from the PSTH to the AUC values due to the models. Recall that the AUC is computed from an ROC analysis on the thresholded $\mu$ function. Here, we define $\mu_{\text{PSTH}}$ to be the estimated firing rate given by the PSTH. Thus, it is the same function for every trial to the repeated stimulus. We compared the AUC values in the same manner as in the model procedure above, building the $\mu_{\text{PSTH}}$ function on 90% of the trials and holding out 10% of the trials for the ROC computation. Figure 3D shows the comparison: for almost every cell the full model is better at predicting the spikes than the PSTH itself, even though the stimulus component of the model is merely a linear filter.

If the extra-stimulus variability has truly been averaged out of the PSTH, the stimulus-only model should do equally well in modeling the PSTH as the full model. To compare the ability for different models to reconstruct the PSTH, we computed the predicted firing rates ($\mu$) to each of the 120 trials of the same white noise movie, and the predicted PSTH is simply the average of these 120 temporal functions. We computed these model predictions for the LFP-only model, stimulus-only model, and full model. Figure 4A shows examples of these simulated PSTHs for these three conditions. Figure 4B shows the overall results for the population. The stimulus model predicted the PSTH

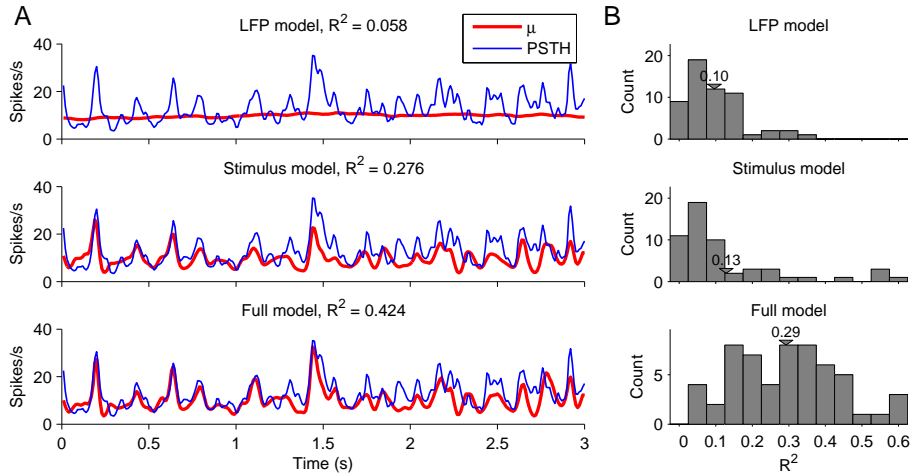

Figure 4: A: For an example cell, the ability for different models to predict the PSTH. Taking the network state into account yields a closer estimate to the PSTH, indicating that the PSTH contains effects unrelated to the stimulus. B: Population histograms of the PSTH variance explained. Including all the terms yields a dramatic increase in the variance explained across the population.

well for some cells, but for most others the stimulus model alone cannot match the full model's performance, indicating a corruption of the PSTH by network effects.

## 5 Conclusions

In this paper we have implemented a L1 regularized point process model to account for stimulus effects, neuronal interactions and network state effects for explaining the spiking activity of V1 neurons. We have showed the derivation for a form of L1 regularized Poisson regression, and identified and implemented a number of computational approaches including coordinate descent and the regularization path. These are crucial for solving the point process model for *in vivo* V1 data, and to our knowledge have not been previously attempted on this scale.

Using this model, we have shown that activity of cells in the surrounding population can account for a significant amount of the variance in the firing of many neurons. We found that the LFP, a broad indicator of the synaptic activity of many cells across a large region (the network state), can account for a large share of these influences from the surrounding cells. This suggests that these spikes are due to the general network state rather than precise spike timing or individual true synaptic connections between a pair of cells. This is consistent with earlier observations that the spiking activity of a neuron is linked to ongoing population activity as measured with optical imaging [12] and LFP [13]. This link to the state of the local population is an influential force affecting the variability in a cell's spiking behavior. Indeed, groups of neurons transition between "Up" (depolarized) and "Down" (hyperpolarized) states, which leads to cycles of higher and lower than normal firing rates (for review, see [14]). These state transitions occur in sleeping and anesthetized animals, in cortical slices [15], as well as in awake animal [16, 17] and awake human patients [18, 19], and might be responsible for generating much of the slow time scale correlation. Our additional experiments showed similar results are found in experiments with natural movie stimulation.

By directly modeling these sources of variability, this method begins to allow us to obtain better encoding models and more accurately isolate the elements of the stimulus that are truly driving the cells' responses. By attributing portions of firing to network state effects (as indicated by the LFP), this approach can obtain more accurate estimates of the underlying connectivity among neurons in cortical circuits.

# References

[1] Jonathon Shlens, Greg D Field, Jeffrey L Gauthier, Martin Greschner, Alexander Sher, Alan M Litke, and E J Chichilnisky. The structure of large-scale synchronized firing in primate retina. *J Neurosci*, 29(15):5022–31, Apr 2009.

[2] Wilson Truccolo, Leigh R Hochberg, and John P Donoghue. Collective dynamics in human and monkey sensorimotor cortex: predicting single neuron spikes. *Nat Neurosci*, 13(1):105–11, Jan 2010.

[3] Jonathan W Pillow, Jonathon Shlens, Liam Paninski, Alexander Sher, Alan M Litke, E J Chichilnisky, and Eero P Simoncelli. Spatio-temporal correlations and visual signalling in a complete neuronal population. *Nature*, 454(7207):995–9, Aug 2008.

[4] Robert E. Kass, Valerie Ventura, and Emory N. Brown. Statistical issues in the analysis of neuronal data. *J Neurophysiol*, 94:8–25, 2005.

[5] Jerome Friedman, Trevor Hastie, and Robert Tibshirani. Regularization paths for generalized linear models via coordinate descent. *Department of Statistics*, Jan 2008.

[6] Mee Young Park and Trevor Hastie. L1 regularization path algorithm for generalized linear models. *Journal of the Royal Statistical Society: Series B (Statistical Methodology)*, 69(4):659–677, 2007.

[7] Nicholas G Hatsopoulos, Qingqing Xu, and Yali Amit. Encoding of movement fragments in the motor cortex. *J Neurosci*, 27(19):5105–14, May 2007.

[8] Matthew A Smith and Adam Kohn. Spatial and temporal scales of neuronal correlation in primary visual cortex. *J Neurosci*, 28(48):12591–603, Nov 2008.

[9] P J Rousche and Richard A Normann. A method for pneumatically inserting an array of penetrating electrodes into cortical tissue. *Ann Biomed Eng*, 20(4):413–22, Jan 1992.

[10] Shy Shoham, Matthew R Fellows, and Richard A Normann. Robust, automatic spike sorting using mixtures of multivariate t-distributions. *J Neurosci Methods*, 127(2):111–22, Aug 2003.

[11] Ryan C Kelly, Matthew A Smith, Jason M Samonds, Adam Kohn, A B Bonds, J Anthony Movshon, and Tai Sing Lee. Comparison of recordings from microelectrode arrays and single electrodes in the visual cortex. *J Neurosci*, 27(2):261–4, Jan 2007.

[12] M Tsodyks, Tal Kenet, Amiram Grinvald, and A Arieli. Linking spontaneous activity of single cortical neurons and the underlying functional architecture. *Science*, 286(5446):1943–6, Dec 1999.

[13] Ian Nauhaus, Laura Busse, Matteo Carandini, and Dario L Ringach. Stimulus contrast modulates functional connectivity in visual cortex. *Nat Neurosci*, 12(1):70–6, Jan 2009.

[14] Alain Destexhe and Diego Contreras. Neuronal computations with stochastic network states. *Science*, 314(5796):85–90, Oct 2006.

[15] Hope A Johnson and Dean V Buonomano. Development and plasticity of spontaneous activity and up states in cortical organotypic slices. *J Neurosci*, 27(22):5915–25, May 2007.

[16] David A Leopold, Yusuke Murayama, and Nikos K Logothetis. Very slow activity fluctuations in monkey visual cortex: implications for functional brain imaging. *Cereb Cortex*, 13(4):422–33, Apr 2003.

[17] Artur Luczak, Peter Barthó, Stephan L Marguet, György Buzsáki, and Kenneth D Harris. Sequential structure of neocortical spontaneous activity in vivo. *Proc Natl Acad Sci USA*, 104(1):347–52, Jan 2007.

[18] Biyu J He, Abraham Z Snyder, John M Zempel, Matthew D Smyth, and Marcus E Raichle. Electrophysiological correlates of the brain's intrinsic large-scale functional architecture. *Proc Natl Acad Sci USA*, 105(41):16039–44, Oct 2008.

[19] Yuval Nir, Roy Mukamel, Ilan Dinstein, Eran Privman, Michal Harel, Lior Fisch, Hagar Gelbard-Sagiv, Svetlana Kipervasser, Fani Andelman, Miri Y Neufeld, Uri Kramer, Amos Arieli, Itzhak Fried, and Rafael Malach. Interhemispheric correlations of slow spontaneous neuronal fluctuations revealed in human sensory cortex. *Nat Neurosci*, 11(9):1100–8, Sep 2008.

